# Weight Space Probability Densities in Stochastic Learning: I. Dynamics and Equilibria

**Todd K. Leen and John E. Moody**
Department of Computer Science and Engineering
Oregon Graduate Institute of Science & Technology
19600 N.W. von Neumann Dr.
Beaverton, OR 97006-1999

## Abstract

The ensemble dynamics of stochastic learning algorithms can be studied using theoretical techniques from statistical physics. We develop the equations of motion for the weight space probability densities for stochastic learning algorithms. We discuss equilibria in the diffusion approximation and provide expressions for special cases of the LMS algorithm. The equilibrium densities are not in general thermal (Gibbs) distributions in the objective function being minimized, but rather depend upon an effective potential that includes diffusion effects. Finally we present an exact analytical expression for the time evolution of the density for a learning algorithm with weight updates proportional to the *sign* of the gradient.

## 1 Introduction: Theoretical Framework

Stochastic learning algorithms involve weight updates of the form

$$\omega(n+1) \;=\; \omega(n) \;+\; \mu(n)\, H[\,\omega(n), x(n)\,] \tag{1}$$

where $\omega \in \mathbb{R}^m$ is the vector of $m$ weights, $\mu$ is the learning rate, $H[\cdot] \in \mathbb{R}^m$ is the update function, and $x(n)$ is the exemplar (input or input/target pair) presented to the network at the $n^{th}$ iteration of the learning rule. Often the update function is based on the gradient of a cost function $H(\omega, x) = -\partial \mathcal{E}(\omega, x)/\partial \omega$. We assume that the exemplars are i.i.d. with underlying probability density $\rho(x)$.

We are interested in studying the time evolution and steady state behavior of the weight space probability density $P(\omega, n)$ for ensembles of networks trained by stochastic learning. Stochastic process theory and classical statistical mechanics provide tools for doing this. As we shall see, the ensemble behavior of stochastic learning algorithms is similar to that of diffusion processes in physical systems, although significant differences do exist.

## 1.1   Dynamics of the Weight Space Probability Density

Equation (1) defines a Markov process on the weight space. Given the *particular input x*, the single time-step transition probability density for this process is a Dirac delta function whose arguments satisfy the weight update (1):

$$W(\omega' \to \omega \mid x) = \delta(\omega - \omega' - \mu H[\omega', x]) \quad . \tag{2}$$

From this conditional transition probability, we calculate the *total* single time-step transition probability (Leen and Orr 1992, Ritter and Schulten 1988)

$$W(\omega' \to \omega) = \langle \delta(\omega - \omega' - \mu H[\omega', x]) \rangle_x \tag{3}$$

where $\langle \ldots \rangle_x$ denotes integration over the measure on the random variable $x$.

The time evolution of the density is given by the Kolmogorov equation

$$P(\omega, n+1) = \int d\omega' \, P(\omega', n) \, W(\omega' \to \omega), \tag{4}$$

which forms the basis for our dynamical description of the weight space probability density [1].

Stationary, or equilibrium, probability distributions are eigenfunctions of the transition probability

$$P_s(\omega) = \int d\omega' \, P_s(\omega') \, W(\omega' \to \omega). \tag{5}$$

It is particularly interesting to note that for problems in which there exists an optimal weight $\omega_*$ such that

$$H(\omega_*, x) = 0, \quad \forall x \quad ,$$

one stationary solution is a delta function at $\omega = \omega_*$. An important class of such examples are noise-free mapping problems for which weight values exist that realize the desired mapping over all possible input/target pairs. For such problems, the ensemble can settle into a sharp distribution at the optimal weights (for examples see Leen and Orr 1992, Orr and Leen 1993).

Although the Kolmogorov equation can be integrated numerically, we would like to make further analytic progress. Towards this end we convert the Kolmogorov

equation into a differential-difference equation by expanding (3) as a power series in $\mu$. Since the transition probability is defined in the sense of generalized functions (i.e. distributions), the proper way to proceed is to smear (4) with a smooth test function of compact support $f(\omega)$ to obtain

$$\int d\omega\, f(\omega)\, P(\omega, n+1) = \int d\omega\, d\omega'\, f(\omega)\, P(\omega', n)\, W(\omega' \to \omega) \ . \qquad (6)$$

Next we use the transition probability (3) to perform the integration over $\omega$ and expand the resulting expression as a power series in $\mu$. Finally, we integrate by parts to take derivatives off $f$, dropping the surface terms. This results in a discrete time version of the classic Kramers-Moyal expansion (Risken 1989)

$$P(\omega, n+1) - P(\omega, n) =$$

$$\sum_{i=1}^{\infty} \frac{(-1)^i}{i!} \sum_{j_1,\dots j_i=1}^{m} \frac{\partial^i}{\partial\omega_{j_1}\,\partial\omega_{j_2}\dots\partial\omega_{j_i}} \left\{ \langle \mu H_{j_1}\, \mu H_{j_2} \dots \mu H_{j_i} \rangle_x\, P(\omega, n) \right\} \ , \quad (7)$$

where $H_{j_a}$ denotes the $j_a{}^{th}$ component of the $m$-component vector $H$.

In section 3, we present an algorithm for which the Kramers-Moyal expansion can be explicitly summed. In general the full expansion is not analytically tractable, and to make further analytic progress we will truncate it at second order to obtain the Fokker-Planck equation.

## 1.2   The Fokker-Planck (Diffusion) Approximation

For small enough $|\mu H|$, the Kramers-Moyal expansion (7) can be truncated to second order to obtain a Fokker-Planck equation:[2]

$$P(\omega, n+1) - P(\omega, n) =$$

$$-\mu \frac{\partial}{\partial\omega_i} [\, A_i(\omega)\, P(\omega, n)\,] + \frac{\mu^2}{2} \frac{\partial^2}{\partial\omega_i\partial\omega_j} [\, B_{ij}(\omega)\, P(\omega, n)\,] \ . \qquad (8)$$

In (8), and throughout the remainder of the paper, repeated indices are summed over. In the Fokker-Planck approximation, only two coefficients appear: $A_i(\omega) \equiv \langle H_i \rangle_x$, called the *drift vector*, and $B_{ij}(\omega) \equiv \langle H_i H_j \rangle_x$, called the *diffusion matrix*. The drift vector is simply the average update applied at $\omega$. Since the diffusion coefficients can be strongly dependent on the position in weight space, the equilibrium densities will, in general, *not be thermal* (Gibbs) distributions in the potential corresponding to $\langle H(\omega, x) \rangle_x$. This is exemplified in our discussion of equilibrium densities for the LMS algorithm in section 2.1 below[3].

## 2    Equilibrium Densities in the Fokker-Planck Approximation

In equilibrium the probability density is stationary, $P(\omega, n+1) = P(\omega, n) \equiv P_s(\omega)$, so the Fokker-Planck equation (8) becomes

$$0 = -\frac{\partial}{\partial \omega_i} J_i(\omega) \equiv -\frac{\partial}{\partial \omega_i} \left( \mu A_i(\omega) P_s(\omega) - \frac{\mu^2}{2} \frac{\partial}{\partial \omega_j} [B_{ij}(\omega) P_s(\omega)] \right) . \quad (9)$$

Here, we have implicitly defined the probability density current $J(\omega)$. In equilibrium, its divergence is zero.

If the drift and diffusion coefficients satisfy *potential conditions*, then the equilibrium current itself is zero and *detailed balance* is obtained. The potential conditions are (Gardiner, 1990)

$$\frac{\partial Z_k}{\partial \omega_l} - \frac{\partial Z_l}{\partial \omega_k} \equiv 0, \quad \text{where} \quad Z_k(\omega) \equiv B_{ki}^{-1}(\omega) \left[ \frac{\mu}{2} \frac{\partial}{\partial \omega_j} B_{ij}(\omega) - A_i(\omega) \right] . \quad (10)$$

Under these conditions the solution to (9) for the equilibrium density is:

$$P_s(\omega) = \frac{1}{K} e^{-2 \mathcal{F}(\omega)/\mu}, \quad \mathcal{F}(\omega) \equiv \int_\omega d\omega_k \, Z_k(\omega) \quad (11)$$

where $K$ is a normalization constant and $\mathcal{F}(\omega)$ is called the *effective potential*.

In general, the potential conditions are not satisfied for stochastic learning algorithms in multiple dimensions.[4] In this respect, stochastic learning differs from most physical diffusion processes. However for LMS with inputs whose correlation matrix is isotropic, the conditions *are* satisfied and the equilibrium density can be reduced to the quadrature in (11).

### 2.1    Equilibrium Density for the LMS Algorithm

The best known on-line learning system is the LMS adaptive filter. For the LMS algorithm, the training examples consist of input/target pairs $x(n) = \{s(n), t(n)\}$, the model output is $u(n) = \omega \cdot s(n)$, and the cost function is the squared error:

$$\mathcal{E}(\omega, x(n)) = \frac{1}{2} [t(n) - u(n)]^2 = \frac{1}{2} [t(n) - \omega \cdot s(n)]^2 . \quad (12)$$

The resulting update equations (for constant learning rate $\mu$) are

$$\omega(n+1) = \omega(n) + \mu [t(n) - \omega \cdot s(n)] s(n). \quad (13)$$

We assume that the training data are generated according to a "signal plus noise" model:

$$t(n) = \omega_* \cdot s(n) + \epsilon(n) , \quad (14)$$

where $\omega_*$ is the "true" weight vector and $\epsilon(n)$ is i.i.d. noise with mean zero and variance $\sigma^2$. We denote the correlation matrix of the inputs $s(n)$ by $R$ and the

fourth order correlation tensor of the inputs by $S$. It is convenient to shift the origin of coordinates in weight space and define the *weight error* vector

$$v \equiv \omega - \omega_*.$$

In terms of $v$, the weight update is

$$v(n+1) = v(n) - \mu \left[ s(n) \cdot v(n) \right] s(n) + \mu \epsilon(n) s(n).$$

The drift vector and diffusion matrix are given by

$$A_i = -\langle s_i s_j \rangle_s v_j = -R_{ij} v_j \tag{15}$$

and

$$B_{ij} = \langle s_i s_j s_k s_l v_k v_l + \epsilon^2 s_i s_j \rangle_{s,\epsilon} = S_{ijkl} v_k v_l + \sigma^2 R_{ij} \tag{16}$$

respectively. Notice that the diffusion matrix is quadratic in $v$. Thus as we move away from the global minimum at $v = 0$, diffusive spreading of the probability density is enhanced. Notice also that, in general, both terms of the diffusion matrix contribute an *anisotropy*.

We further assume that the inputs are drawn from a zero-mean Gaussian process. This assumption allows us to appeal to the Gaussian moment factoring theorem (Haykin, 1991, p318) to express the fourth-order correlation $S$ in terms of $R$

$$S_{ijkl} = R_{ij} R_{kl} + R_{ik} R_{jl} + R_{il} R_{jk} \ .$$

The diffusion matrix reduces to

$$B = (v^T R v + \sigma^2) R + 2(Rv)(Rv)^T \ . \tag{17}$$

To compute the effective potential (10 and 11) the diffusion matrix is inverted using the Sherman-Morrison formula (Press, 1987, p67). As a final simplification, we assume that the input distribution is spherically symmetric. Thus

$$R = r I \ ,$$

where $I$ denotes the identity matrix.

Together these assumptions insure detailed balance, and we can integrate (11) in closed form. In figure 1, we compare the effective potential $\mathcal{F}(v)$ (for 1-D LMS) with the potential corresponding to the quadratic cost function.

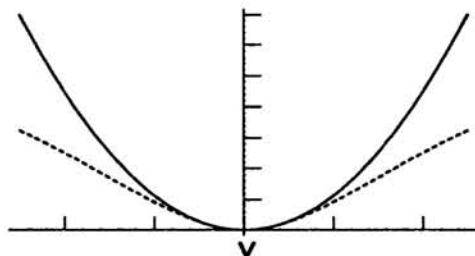

Fig.1: Effective potential (dashed curve) and cost function (solid curve) for 1-D LMS.

The spatial dependence of the the diffusion coefficient forces the effective potential to soften relative to the cost function for large $|v|$. This accentuates the tails of the distribution relative to a gaussian.

The equilibrium density is

$$P_s(v) = \frac{1}{K} \left[ 1 + \frac{3r}{\sigma^2} |v|^2 \right]^{-\left(\frac{2+m}{3} + \frac{1}{3r\mu}\right)}, \tag{18}$$

where, as before, $m$ and $K$ denote the dimension of the weight vector and the normalization constant for the density respectively. For a *1-D filter*, the equilibrium density can be found in closed form without assuming Gaussian input data. We find

$$P_s(v) = \frac{1}{K} \left[ 1 + \frac{S}{r\,\sigma^2} v^2 \right]^{-\left(1 + \frac{r}{\mu S}\right)}. \tag{19}$$

With gaussian inputs (for which $S = 3r^2$) (19) properly reduces to (18) with $m = 1$.

The equilibrium densities (18) and (19) are clearly *not* gaussian, however in the limit of very small $\mu r$ they reduce to gaussian distributions with variance $\mu\sigma^2/2$. Figure 2 shows a comparison between the theoretical result and a histogram of 200,000 values of $v$ generated by simulation with $\mu = 0.005$, and $\sigma^2 = 1.0$. The input data were drawn from a zero-mean Gaussian distribution with $r = 4.0$.

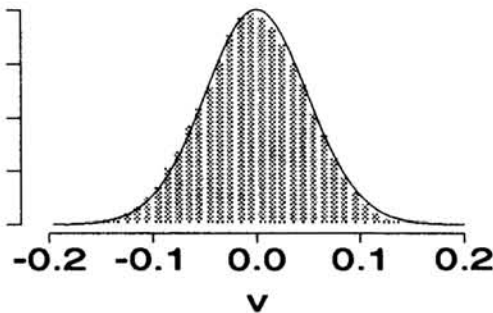

Fig.2: Equilibrium density for 1-D LMS

## 3   An Exactly Summable Model

As in the case of LMS learning above, stochastic gradient descent algorithms update weights based on an instantaneous estimate of the gradient of some average cost function $\mathcal{E}(\omega) = \langle \mathcal{E}(\omega, x) \rangle_x$. That is, the update is given by

$$H_i(\omega, x) = -\frac{\partial}{\partial \omega_i} \mathcal{E}(\omega, x).$$

An alternative is to increment or decrement each weight by a fixed amount depending only on the *sign* of $\partial \mathcal{E}/\partial \omega_i$. We formulated this alternative update rule because it avoids a common problem for sigmoidal networks, getting stuck on "flat spots" or "plateaus". The standard gradient descent update rule yields very slow movement on plateaus, while second order methods such as gauss-newton can be unstable. The sign-of-gradient update rule suffers from neither of these problems.[5]

If at each iteration one chooses a weight at random for updating, then the Kramers-Moyal expansion can be exactly summed. Thus at each iteration we 1) choose a weight $\omega_i$ and an exemplar $x$ at random, and 2) update $\omega_i$ with

$$H_i(\omega, x) \; = \; -\operatorname{sign}\left(\frac{\partial \mathcal{E}(\omega, x(n))}{\partial \omega_i}\right) \; . \tag{20}$$

With this update rule, $H_j = \pm 1$ or $0$ and $H_i H_j = \delta_{ij}$ (or $0$). All of the coefficients $\langle H_i H_j H_k \dots \rangle_x$ in the Kramers-Moyal expansion (7) vanish *unless* $i = j = k = \dots$. The remaining series can be summed by breaking it into odd and even parts. This leaves

$$P(\omega, n+1) \; - \; P(\omega, n) \; =$$

$$- \; \frac{1}{2m} \sum_{j=1}^{m} \{\, P(\omega + \mu_j, n)\, A_j(\omega + \mu_j) - P(\omega - \mu_j, n)\, A_j(\omega - \mu_j)\,\}$$

$$+ \; \frac{1}{2m} \sum_{j=1}^{m} \{\, P(\omega + \mu_j, n)\, B_{jj}(\omega + \mu_j) - 2\, P(\omega, n)\, B_{jj}(\omega)$$

$$+ \; P(\omega - \mu_j, n)\, B_{jj}(\omega - \mu_j)\,\} \tag{21}$$

where $\mu_j$ denotes a displacement along $\omega_j$ a distance $\mu$, $A_j(\omega) \equiv \langle H_j(\omega, x) \rangle_x$, and $B_{jj}(\omega) \equiv \langle H_j^2(\omega, x) \rangle_x$. Note that $B_{jj}(\omega) = 1$ unless $H(\omega, x) = 0$, for all $x$, in which case $B_{jj}(\omega) \equiv 0$. Although exact, (21) curiously has the form of a second order finite difference approximation to the Fokker-Planck equation with diagonal diffusion matrix. This form is understandable, since the dynamics (20) restrict the weight values $\omega$ to a hypercubic lattice with cell length $\mu$ and generate only nearest neighbor interactions.

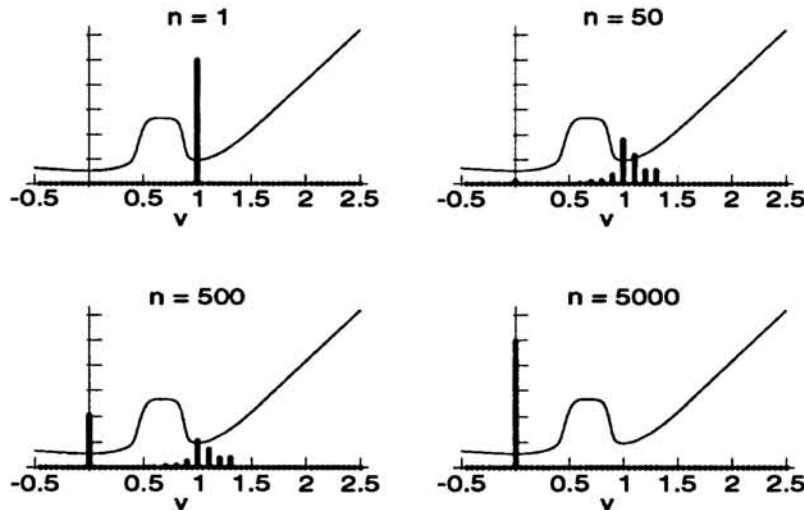

Fig.3: Sequence of densities for the XOR problem

As an example, figure 3 shows the cost function evaluated along a 1-D slice through the weight space for the XOR problem. Along this line are local and global minima at $v = 1$ and $v = 0$ respectively. Also shown is the probability density (vertical lines). The sequence shows the spreading of the density from its initialization at the local minimum, and its eventual collection at the global minimum.

## 4  Discussion

A theoretical approach that focuses on the dynamics of the weight space probability density, as we do here, provides powerful tools to extend understanding of stochastic search. Both transient and equilibrium behavior can be studied using these tools. We expect that knowledge of equilibrium weight space distributions can be used in conjunction with theories of generalization (e.g. Moody, 1992) to assess the influence of stochastic search on prediction error. Characterization of transient phenomena should facilitate the design and evaluation of search strategies such as data batching and adaptive learning rate schedules. Transient phenomena are treated in greater depth in the companion paper in this volume (Orr and Leen, 1993).

### Acknowledgements

T. Leen was supported under grants N00014-91-J-1482 and N00014-90-J-1349 from ONR. J. Moody was supported under grants 89-0478 from AFOSR, ECS-9114333 from NSF, and N00014-89-J-1228 and N00014-92-J-4062 from ONR.

## Footnotes

[1]An alternative is to base the time evolution on a suitable master equation. Both approaches give the same results.

[2]Radons *et al.* (1990) independently derived a Fokker-Planck equation for backpropagation. Earlier, Ritter and Schulten (1988) derived a Fokker-Planck equation (for Kohonen's self-ordering feature map) that is valid in the neighborhood of a local optimum.

[3]See (Leen and Orr 1992, Orr and Leen 1993) for further examples.

[4] For one-dimensional algorithms, the potential conditions are *trivially* satisfied.

[5]The use of the sign of the gradient has been suggested previously in the stochastic approximation literature by Fabian (1960) and in the neural network literature by Derthick (1984).

### References

Todd K. Leen and Genevieve B. Orr (1992), Weight-space probability densities and convergence times for stochastic learning. In *International Joint Conference on Neural Networks*, pages IV 158–164. IEEE, June.

H. Ritter and K. Schulten (1988), Convergence properties of Kohonen's topology conserving maps: Fluctuations, stability and dimension selection, *Biol. Cybern.*, 60, 59-71.

Genevieve B. Orr and Todd K. Leen (1993), Probability densities in stochastic learning: II. Transients and Basin Hopping Times. In Giles, C.L., Hanson, S.J., and Cowan, J.D. (eds.), *Advances in Neural Information Processing Systems 5*. San Mateo, CA: Morgan Kaufmann Publishers.

H. Risken (1989), *The Fokker-Planck Equation* Springer-Verlag, Berlin.

G. Radons, H.G. Schuster and D. Werner (1990), Fokker-Planck description of learning in backpropagation networks, *International Neural Network Conference – INNC 90*, Paris, II 993-996, Kluwer Academic Publishers.

C.W. Gardiner (1990), *Handbook of Stochastic Methods, 2nd Ed.* Springer-Verlag, Berlin.

Simon Haykin (1991), *Adaptive Filter Theory, 2nd edition.* Prentice Hall, Englewood Cliffs, N.J.

W.H. Press, B.P. Flannery, S.A. Teukolsky, and W.T. Vetterling (1987) *Numerical Recipes – the Art of Scientific Computing.* Cambridge University Press, Cambridge / New York.

V. Fabian (1960), Stochastic approximation methods. *Czechoslovak Math J.*, 10, 123–159.

Mark Derthick (1984), Variations on the Boltzmann machine learning algorithm. Technical Report CMU-CS-84-120, Department of Computer Science, Carnegie-Mellon University, Pittsburgh, PA, August.

John E. Moody (1992), The effective number of parameters: An analysis of generalization and regularization in nonlinear learning systems. In J.E. Moody, S.J. Hanson, and R.P. Lipmann, editors, *Advances in Neural Information Processing Systems 4*. Morgan Kaufmann Publishers, San Mateo, CA.